# Convergence Rate Analysis of MAP Coordinate Minimization Algorithms

**Ofer Meshi** [*]
meshi@cs.huji.ac.il

**Tommi Jaakkola** [†]
tommi@csail.mit.edu

**Amir Globerson** [*]
gamir@cs.huji.ac.il

## Abstract

Finding maximum a posteriori (MAP) assignments in graphical models is an important task in many applications. Since the problem is generally hard, linear programming (LP) relaxations are often used. Solving these relaxations efficiently is thus an important practical problem. In recent years, several authors have proposed message passing updates corresponding to coordinate descent in the dual LP. However, these are generally not guaranteed to converge to a global optimum. One approach to remedy this is to smooth the LP, and perform coordinate descent on the smoothed dual. However, little is known about the convergence rate of this procedure. Here we perform a thorough rate analysis of such schemes and derive primal and dual convergence rates. We also provide a simple dual to primal mapping that yields feasible primal solutions with a guaranteed rate of convergence. Empirical evaluation supports our theoretical claims and shows that the method is highly competitive with state of the art approaches that yield global optima.

## 1   Introduction

Many applications involve simultaneous prediction of multiple variables. For example, we may seek to label pixels in an image, infer amino acid residues in protein design, or find the semantic role of words in a sentence. These problems can be cast as maximizing a function over a set of labels (or minimizing an energy function). The function typically decomposes into a sum of local functions over overlapping subsets of variables.

Such maximization problems are nevertheless typically hard. Even for simple decompositions (e.g., subsets correspond to pairs of variables), maximizing over the set of labels is often provably NP-hard. One approach would be to reduce the problem to a tractable one, e.g., by constraining the model to a low tree-width graph. However, empirically, using more complex interactions together with approximate inference methods is often advantageous. One popular family of approximate methods is the linear programming (LP) relaxation approach. Although these LPs are generally tractable, general purpose LP solvers typically do not exploit the problem structure [28]. Therefore a great deal of effort has gone into designing solvers that are specifically tailored to typical MAP-LP relaxations. These include, for example, cut based algorithms [2], accelerated gradient methods [8], and augmented Lagrangian methods [10, 12]. One class of particularly simple algorithms, which we will focus on here, are coordinate minimization based approaches. Examples include max-sum-diffusion [25], MPLP [5] and TRW-S [9]. These work by first taking the dual of the LP and then optimizing the dual in a block coordinate fashion [21]. In many cases, the coordinate block operations can be performed in closed form, resulting in updates quite similar to the max-product message passing algorithm. By coordinate minimization we mean that at each step a set of coordinates is chosen, all other coordinates are fixed, and the chosen coordinates are set to their

---

[*]School of Computer Science and Engineering, The Hebrew University of Jerusalem, Israel

[†]CSAIL, MIT, Cambridge, MA

optimal value given the rest. This is different from a coordinate descent strategy where instead a gradient step is performed on the chosen coordinates (rather than full optimization).

A main caveat of the coordinate minimization approach is that it will not necessarily find the global optimum of the LP (although in practice it often does). This is a direct result of the LP objective not being strictly convex. Several authors have proposed to smooth the LP with entropy terms and employ variants of coordinate minimization [7, 26]. However, the convergence rates of these methods have not been analyzed. Moreover, since the algorithms work in the dual, there is no simple procedure to map the result back into primal feasible variables. We seek to address all these shortcomings: we present a convergence rate analysis of dual coordinate minimization algorithms, provide a simple scheme for generating primal variables from dual ones, and analyze the resulting primal convergence rates.

Convergence rates for coordinate minimization are not common in the literature. While asymptotic convergence is relatively well understood [22], finite rates have been harder to obtain. Recent work [17] provides rates for rather limited settings which do not hold in our case. On the other hand, for coordinate descent methods, some rates have been recently obtained for greedy and stochastic update schemes [16, 20]. These do not apply directly to the full coordinate minimization case which we study. A related analysis of MAP-LP using smoothing appeared in [3]. However, their approach is specific to LDPC codes, and does not apply to general MAP problems as we analyze here.

## 2 MAP and LP relaxations

Consider a set of $n$ discrete variables $x_1, \ldots, x_n$, and a set $C$ of subsets of these variables (i.e., $c \in C$ is a subset of $\{1, \ldots, n\}$). We consider maximization problems over functions that decompose according to these subsets. In particular, each subset $c$ is associated with a local function or factor $\theta_c(x_c)$ and we also include factors $\theta_i(x_i)$ for each individual variable.[1] The MAP problem is to find an assignment $x = (x_1, \ldots, x_n)$ to all the variables which maximizes the sum of these factors:

$$\text{MAP}(\theta) = \max_x \sum_{c \in C} \theta_c(x_c) + \sum_{i=1}^{n} \theta_i(x_i) \tag{1}$$

Linear programming relaxations are a popular approach to approximating combinatorial optimization problems [6, 23, 25]. For example, we obtain a relaxation of the discrete optimization problem given in Eq. (1) by replacing it with the following linear program:[2]

$$PMAP: \max_{\mu \in \mathcal{M}_L} P(\mu) = \max_{\mu \in \mathcal{M}_L} \left\{ \sum_c \sum_{x_c} \theta_c(x_c)\mu_c(x_c) + \sum_i \sum_{x_i} \theta_i(x_i)\mu_i(x_i) \right\} = \max_{\mu \in \mathcal{M}_L} \mu \cdot \theta \tag{2}$$

where $P(\mu)$ is the primal (linear) objective and the *local marginal polytope* $\mathcal{M}_L$ enforces basic consistency constraints on the marginals $\{\mu_i(x_i), \forall x_i\}$ and $\{\mu_c(x_c), \forall x_c\}$. Specifically,

$$\mathcal{M}_L = \left\{ \mu \geq 0 : \begin{array}{ll} \sum_{x_{c \backslash i}} \mu_c(x_c) = \mu_i(x_i) & \forall c, i \in c, x_i \\ \sum_{x_i} \mu_i(x_i) = 1 & \forall i \end{array} \right\} \tag{3}$$

If the maximizer of $PMAP$ has only integral values (i.e., 0 or 1) it can be used to find the MAP assignment (e.g., by taking the $x_i$ that maximizes $\mu_i(x_i)$). However, in the general case the solution may be fractional [24] and the maximum of $PMAP$ is an upper bound on $\text{MAP}(\theta)$.

### 2.1 Smoothing the LP

As mentioned earlier, several authors have considered a smoothed version of the LP in Eq. (2). As we shall see, this offers several advantages over solving the LP directly. Given a smoothing parameter $\tau > 0$, we consider the following smoothed primal problem:

$$PMAP_\tau: \quad \max_{\mu \in \mathcal{M}_L} P_\tau(\mu) = \max_{\mu \in \mathcal{M}_L} \left\{ \mu \cdot \theta + \frac{1}{\tau} \sum_c H(\mu_c) + \frac{1}{\tau} \sum_i H(\mu_i) \right\} \tag{4}$$

where $H(\mu_c)$ and $H(\mu_i)$ are local entropy terms. Note that as $\tau \to \infty$ we obtain the original primal problem. In fact, a stronger result can be shown. Namely, that the optimal value of $PMAP$ is $O(\frac{1}{\tau})$ close to the optimal value of $PMAP_\tau$. This justifies using the smoothed objective $P_\tau$ as a proxy to $P$ in Eq. (2). We express this in the following lemma (which appears in similar forms in [7, 15]).

**Lemma 2.1.** *Denote by $\mu^*$ the optimum of problem $PMAP$ in Eq. (2) and by $\hat{\mu}^*$ the optimum of problem $PMAP_\tau$ in Eq. (4). Then:*

$$\hat{\mu}^* \cdot \theta \le \mu^* \cdot \theta \le \hat{\mu}^* \cdot \theta + \frac{H_{\max}}{\tau} \tag{5}$$

*where $H_{\max} = \sum_c \log |x_c| + \sum_i \log |x_i|$. In other words, the smoothed optimum is an $O(\frac{1}{\tau})$-optimal solution of the original non-smoothed problem.*

We shall be particularly interested in the dual of $PMAP_\tau$ since it facilitates simple coordinate minimization updates. Our dual variables will be denoted by $\delta_{ci}(x_i)$, which can be interpreted as the messages from subset $c$ to node $i$ about the value of variable $x_i$. The dual variables are therefore indexed by $(c, i, x_i)$ and written as $\delta_{ci}(x_i)$. The dual objective can be shown to be:

$$F(\delta) = \sum_c \frac{1}{\tau} \log \sum_{x_c} \exp \left( \tau \theta_c(x_c) - \tau \sum_{i:i \in c} \delta_{ci}(x_i) \right) + \sum_i \frac{1}{\tau} \log \sum_{x_i} \exp \left( \tau \theta_i(x_i) + \tau \sum_{c:i \in c} \delta_{ci}(x_i) \right) \tag{6}$$

The dual problem is an unconstrained smooth minimization problem:

$$DMAP_\tau : \min_\delta F(\delta) \tag{7}$$

Convex duality implies that the optima of $DMAP_\tau$ and $PMAP_\tau$ coincide.

Finally, we shall be interested in transformations between dual variables $\delta$ and primal variables $\mu$ (see Section 5). The following are the transformations obtained from the Lagrangian derivation (i.e., they can be used to switch from *optimal* dual variables to *optimal* primal variables).

$$\mu_c(x_c; \delta) \propto \exp \left( \tau \theta_c(x_c) - \tau \sum_{i:i \in c} \delta_{ci}(x_i) \right) \quad , \quad \mu_i(x_i; \delta) \propto \exp \left( \tau \theta_i(x_i) + \tau \sum_{c:i \in c} \delta_{ci}(x_i) \right) \tag{8}$$

We denote the vector of all such marginals by $\mu(\delta)$. For the dual variables $\delta$ that minimize $F(\delta)$ it holds that $\mu(\delta)$ are feasible (i.e., $\mu(\delta) \in \mathcal{M}_L$). However, we will also consider $\mu(\delta)$ for non optimal $\delta$, and show how to obtain primal feasible approximations from $\mu(\delta)$. These will be helpful in obtaining primal convergence rates.

It is easy to see that: $(\nabla F(\delta^t))_{c,i,x_i} = \mu_i(x_i; \delta^t) - \mu_c(x_i; \delta^t)$, where (with some abuse of notation) we denote: $\mu_c(x_i) = \sum_{x_{c \setminus i}} \mu_c(x_{c \setminus i}, x_i)$. The elements of the gradient thus correspond to inconsistency between the marginals $\mu(\delta^t)$ (i.e., the degree to which they violate the constraints in Eq. (3)). We shall make repeated use of this fact to link primal and dual variables.

## 3 Coordinate Minimization Algorithms

In this section we propose several coordinate minimization procedures for solving $DMAP_\tau$ (Eq. (7)). We first set some notation to define block coordinate minimization algorithms. Denote the objective we want to minimize by $F(\delta)$ where $\delta$ corresponds to a set of $N$ variables. Now define $\mathcal{S} = \{S_1, \ldots, S_M\}$ as a set of subsets, where each subset $S_i \subseteq \{1, \ldots, N\}$ describes a coordinate block. We will assume that $S_i \cap S_j = \emptyset$ for all $i, j$ and that $\cup_i S_i = \{1, \ldots, N\}$.

Block coordinate minimization algorithms work as follows: at each iteration, first set $\delta^{t+1} = \delta^t$. Next choose a block $S_i$ and set:

$$\delta_{S_i}^{t+1} = \arg \min_{\delta_{S_i}} F_i(\delta_{S_i}; \delta^t) \tag{9}$$

where we use $F_i(\delta_{S_i}; \delta^t)$ to denote the function $F$ restricted to the variables $\delta_{S_i}$ and where all other variables are set to their value in $\delta^t$. In other words, at each iteration we fully optimize only over the variables $\delta_{S_i}$ while fixing all other variables. We assume that the minimization step in Eq. (9) can be solved in closed form, which is indeed the case for the updates we consider.

Regarding the choice of an update schedule, several options are available:

- **Cyclic**: Decide on a fixed order (e.g., $S_1, \ldots, S_M$) and cycle through it.
- **Stochastic**: Draw an index $i$ uniformly at random[3] at each iteration and use the block $S_i$.
- **Greedy**: Denote by $\nabla_{S_i} F(\delta^t)$ the gradient $\nabla F(\delta^t)$ evaluated at coordinates $S_i$ only. The greedy scheme is to choose $S_i$ that maximizes $\|\nabla_{S_i} F(\delta^t)\|_\infty$. In other words, choose the set of coordinates that correspond to maximum gradient of the function $F$. Intuitively this corresponds to choosing the block that promises the maximal (local) decrease in objective. Note that to find the best coordinate we presumably must process all sets $S_i$ to find the best one. We will show later that this can be done rather efficiently in our case.

In our analysis, we shall focus on the Stochastic and Greedy cases, and analyze their rate of convergence. The cyclic case is typically hard to analyze, with results only under multiple conditions which do not hold here (e.g., see [17]).

Another consideration when designing coordinate minimization algorithms is the choice of block size. One possible choice is all variables $\delta_{ci}(\cdot)$ (for a specific pair $ci$). This is the block chosen in the max-sum-diffusion (MSD) algorithm (see [25] and [26] for non-smooth and smooth MSD). A larger block that also facilitates closed form updates is the set of variables $\delta_{\cdot i}(\cdot)$. Namely, all messages into a variable $i$ from $c$ such that $i \in c$. We call this a *star* update. The update is used in [13] for the non-smoothed dual (but the possibility of applying it to the smoothed version is mentioned).

For simplicity, we focus here only on the star update, but the derivation is similar for other choices. To derive the star update around variable $i$, one needs to fix all variables except $\delta_{\cdot i}(\cdot)$ and then set the latter to minimize $F(\delta)$. Since $F(\delta)$ is differentiable this is pretty straightforward. The update turns out to be:[4]

$$\delta_{ci}^{t+1}(x_i) = \delta_{ci}^t(x_i) + \frac{1}{\tau} \log \mu_c^t(x_i) - \frac{1}{N_i+1} \cdot \frac{1}{\tau} \log \left( \mu_i^t(x_i) \cdot \prod_{c':i\in c'} \mu_{c'}^t(x_i) \right) \qquad (10)$$

where $N_i = |\{c : i \in c\}|$. It is interesting to consider the improvement in $F(\delta)$ as a result of the star update. It can be shown to be exactly:

$$F(\delta^t) - F(\delta^{t+1}) = -\frac{1}{\tau} \log \left( \sum_{x_i} \left( \mu_i^t(x_i) \cdot \prod_{c:i\in c} \mu_c^t(x_i) \right)^{\frac{1}{N_i+1}} \right)^{N_i+1}$$

The RHS is known as *Matusita's divergence measure* [11], and is a generalization of the Bhattacharyya divergence to several distributions. Thus the improvement can be easily computed before actually applying the update and is directly related to how consistent the $N_i + 1$ distributions $\mu_c^t(x_i), \mu_i^t(x_i)$ are. Recall that at the optimum they all agree as $\mu \in \mathcal{M}_L$, and thus the expected improvement is zero.

## 4 Dual Convergence Rate Analysis

We begin with the convergence rates of the dual $F$ using greedy and random schemes described in Section 3. In Section 5 we subsequently show how to obtain a primal feasible solution and how the dual rates give rise to primal rates. Our analysis builds on the fact that we can lower bound the improvement at each step, as a function of some norm of the block gradient.

### 4.1 Greedy block minimization

**Theorem 4.1.** *Define $B_1$ to be a constant such that $\|\delta^t - \delta^*\|_1 \leq B_1$ for all $t$. If coordinate minimization of each block $S_i$ satisfies:*

$$F(\delta^t) - F(\delta^{t+1}) \geq \frac{1}{k} \|\nabla_{S_i} F(\delta^t)\|_\infty^2 \qquad (11)$$

*for all $t$, then for any $\epsilon > 0$ after $T = \frac{kB_1^2}{\epsilon}$ iterations of the greedy algorithm, $F(\delta^T) - F(\delta^*) \leq \epsilon$.*

*Proof.* Using Hölder's inequality we obtain the bound:

$$F(\delta^t) - F(\delta^*) \le \nabla F(\delta^t)^\top (\delta^t - \delta^*) \le \|\nabla F(\delta^t)\|_\infty \cdot \|\delta^t - \delta^*\|_1$$

(12)

Implying: $\|\nabla F(\delta^t)\|_\infty \ge \frac{1}{B_1}(F(\delta^t) - F(\delta^*))$. Now, using the condition on the improvement and the greedy nature of the update, we obtain a bound on the improvement:

$$
\begin{aligned}
F(\delta^t) - F(\delta^{t+1}) &\ge \frac{1}{k}\|\nabla_{S_i} F(\delta^t)\|_\infty^2 = \frac{1}{k}\|\nabla F(\delta^t)\|_\infty^2 \\
&\ge \frac{1}{kB_1^2}\left(F(\delta^t) - F(\delta^*)\right)^2 \ge \frac{1}{kB_1^2}\left(F(\delta^t) - F(\delta^*)\right)\left(F(\delta^{t+1}) - F(\delta^*)\right)
\end{aligned}
$$

Hence,

$$\frac{1}{kB_1^2} \le \frac{F(\delta^t) - F(\delta^*) - \left(F(\delta^{t+1}) - F(\delta^*)\right)}{\left(F(\delta^t) - F(\delta^*)\right)\left(F(\delta^{t+1}) - F(\delta^*)\right)} = \frac{1}{F(\delta^{t+1}) - F(\delta^*)} - \frac{1}{F(\delta^t) - F(\delta^*)} \quad (13)$$

Summing over $t$ we obtain:

$$\frac{T}{kB_1^2} \le \frac{1}{F(\delta^T) - F(\delta^*)} - \frac{1}{F(\delta^0) - F(\delta^*)} \le \frac{1}{F(\delta^T) - F(\delta^*)} \quad (14)$$

and the desired result follows. □

## 4.2 Stochastic block minimization

**Theorem 4.2.** *Define $B_2$ to be a constant such that $\|\delta^t - \delta^*\|_2 \le B_2$ for all t. If coordinate minimization of each block $S_i$ satisfies:*

$$F(\delta^t) - F(\delta^{t+1}) \ge \frac{1}{k}\|\nabla_{S_i} F(\delta^t)\|_2^2 \quad (15)$$

*for all t, then for any $\epsilon > 0$ after $T = \frac{k|\mathcal{S}|B_2^2}{\epsilon}$ iterations of the stochastic algorithm we have that $\mathbb{E}[F(\delta^T)] - F(\delta^*) \le \epsilon$.*[5]

The proof is similar to Nesterov's analysis (see Theorem 1 in [16]). The proof in [16] relies on the improvement condition in Eq. (15) and not on the precise nature of the update. Note that since the cost of the update is roughly linear in the size of the block then this bound does not tell us which block size is better (the cost of an update times the number of blocks is roughly constant).

## 4.3 Analysis of $DMAP_\tau$ block minimization

We can now obtain rates for our coordinate minimization scheme for optimizing $DMAP_\tau$ by finding the $k$ to be used in conditions Eq. (15) and Eq. (11). The result for the star update is given below.

**Proposition 4.3.** *The star update for $x_i$ satisfies the conditions in Eqs. 15 and 11 with $k = 4\tau N_i$.*

This can be shown using Equation 2.4 in [14], which states that if $F_i(\delta_{S_i}; \delta)$ (see Eq. (9)) has Lipschitz constant $L_i$ then Eq. (15) is satisfied with $k = 2L_i$. We can then use the fact that the Lipschitz constant of a star block is at most $2\tau N_i$ (this can be calculated as in [18]) to obtain the result.[6] To complete the analysis, it turns out that $B_1$ and $B_2$ can be bounded via a function of $\theta$ by bounding $\|\delta\|_1$ (see supplementary, Lemma 1.2). We proceed to discuss the implications of these bounds.

## 4.4 Comparing the different schemes

The results we derived have several implications. First, we see that both stochastic and greedy schemes achieve a rate of $O(\frac{\tau}{\epsilon})$. This matches the known rates for regular (non-accelerated) gradient descent on functions with Lipschitz continuous gradient (e.g., see [14]), although in practice coordinate minimization is often much faster.

The main difference between the greedy and stochastic rates is that the factor $|\mathcal{S}|$ (the number of blocks) does not appear in the greedy rate, and does appear in the stochastic one. This can have a considerable effect since $|\mathcal{S}|$ is either the number of variables $n$ (in the star update) or the number of factors $|C|$ (in MPLP). Both can be significant (e.g., $|C|$ is the number of edges in a pairwise MRF model). The greedy algorithm does pay a price for this advantage, since it has to find the optimal block to update at each iteration. However, for the problem we study here this can be done much more efficiently using a priority queue. To see this, consider the star update. A change in the variables $\delta_{\cdot i}(\cdot)$ will only affect the blocks that correspond to variables $j$ that are in $c$ such that $i \in c$. In many cases this is small (e.g., low degree pairwise MRFs) and thus we will only have to change the priority queue a small number of times, and this cost would be negligible when using a Fibonacci heap for example.[7] Indeed, our empirical results show that the greedy algorithm consistently outperforms the stochastic one (see Section 6).

## 5   Primal convergence

Thus far we have considered only dual variables. However, it is often important to recover the primal variables. We therefore focus on extracting primal feasible solutions from current $\delta$, and characterize the degree of primal optimality and associated rates. The primal variables $\mu(\delta)$ (see Eq. (8)) need not be feasible in the sense that the consistency constraints in Eq. (3) are not necessarily satisfied. This is true also for other approaches to recovering primal variables from the dual, such as averaging subgradients when using subgradient descent (see, e.g., [21]).

We propose a simple two-step algorithm for transforming any dual variables $\delta$ into primal feasible variables $\tilde{\mu}(\delta) \in \mathcal{M}_L$. The resulting $\tilde{\mu}(\delta)$ will also be shown to converge to the optimal primal solution in Section 5.1. The procedure is described in Algorithm 1 below.

---

**Algorithm 1** Mapping to feasible primal solution

---

**Step 1:** Make marginals consistent.

For all $i$ do: $\bar{\mu}_i(x_i) = \frac{1}{1+\sum_{c:i \in c} \frac{1}{|X_{c \backslash i}|}} \left( \mu_i(x_i) + \sum_{c:i \in c} \frac{1}{|X_{c \backslash i}|} \mu_c(x_i) \right)$

For all $c$ do: $\bar{\mu}_c(x_c) = \mu_c(x_c) - \sum_{i:i \in c} \frac{1}{|X_{c \backslash i}|} \left( \mu_c(x_i) - \bar{\mu}_i(x_i) \right)$

**Step 2:** Make marginals non-negative.

$\overline{\lambda = 0}$

**for** $c \in C, x_c$ **do**

    **if** $\bar{\mu}_c(x_c) < 0$ **then**

        $\lambda = \max \left\{ \lambda, \frac{-\bar{\mu}_c(x_c)}{-\bar{\mu}_c(x_c)+\frac{1}{|X_c|}} \right\}$

    **else if** $\bar{\mu}_c(x_c) > 1$ **then**

        $\lambda = \max \left\{ \lambda, \frac{\bar{\mu}_c(x_c)-1}{\bar{\mu}_c(x_c)-\frac{1}{|X_c|}} \right\}$

    **end if**

**end for**

**for** $\ell = 1, \ldots, n; c \in C$ **do**

    $\tilde{\mu}_\ell(x_\ell) = (1-\lambda)\bar{\mu}_\ell(x_\ell) + \lambda \frac{1}{|X_\ell|}$

**end for**

---

Importantly, all steps consist of cheap elementary local calculations in contrast to other methods previously proposed for this task (compare to [18, 27]). The first step performs a Euclidian projection of $\mu(\delta)$ to consistent marginals $\bar{\mu}$. Specifically, it solves:

$$\min_{\bar{\mu}} \quad \frac{1}{2}\|\mu(\delta) - \bar{\mu}\|^2 \quad , \quad \text{s.t. } \bar{\mu}_c(x_i) = \bar{\mu}_i(x_i), \text{ for all } c, i \in c, x_i \quad , \quad \sum_i \bar{\mu}_i(x_i) = 1, \text{ for all } i$$

Note that we did not include non-negativity constraints above, so the projection might result in negative $\bar{\mu}$. In the second step we "pull" $\bar{\mu}$ back into the feasible regime by taking a convex combination

with the uniform distribution $u$ (see [3] for a related approach). In particular, this step solves the simple problem of finding the smallest $\lambda \in [0, 1]$ such that $0 \leq \tilde{\mu} \leq 1$ (where $\tilde{\mu} = (1 - \lambda)\bar{\mu} + \lambda u$). Since this step interpolates between two distributions that satisfy consistency and normalization constraints, $\tilde{\mu}$ will be in the local polytope $\mathcal{M}_L$.

## 5.1 Primal convergence rate

Now that we have a procedure for obtaining a primal solution we analyze the corresponding convergence rate. First, we show that if we have $\delta$ for which $\|\nabla F(\delta)\|_\infty \leq \epsilon$ then $\tilde{\mu}(\delta)$ (after Algorithm 1) is an $O(\epsilon)$ primal optimal solution.

**Theorem 5.1.** *Denote by $P_\tau^*$ the optimum of the smoothed primal $PMAP_\tau$. For any set of dual variables $\delta$, and any $\epsilon \in R(\tau)$ (see supp. for definition of $R(\tau)$) it holds that if $\|\nabla F(\delta)\|_\infty \leq \epsilon$ then $P_\tau^* - P_\tau(\tilde{\mu}(\delta)) \leq C_0 \epsilon$. The constant $C_0$ depends only on the parameters $\theta$ and is independent of $\tau$.*

The proof is given in the supplementary file (Section 1). The key idea is to break $F(\delta) - P_\tau(\tilde{\mu}(\delta))$ into components, and show that each component is upper bounded by $O(\epsilon)$. The range $R(\tau)$ consists of $\epsilon \geq O(\frac{1}{\tau})$ and $\epsilon \leq O(e^{-\tau})$. As we show in the supplementary this range is large enough to guarantee any desired accuracy in the non-smoothed primal. We can now translate dual rates into primal rates. This can be done via the following well known lemma:

**Lemma 5.2.** *Any convex function $F$ with Lipschitz continuous gradient and Lipschitz constant $L$ satisfies $\|\nabla F(\delta)\|_2^2 \leq 2L \left( F(\delta) - F(\delta^*) \right)$.*

These results together with the fact that $\|\nabla F(\delta)\|_2^2 \geq \|\nabla F(\delta)\|_\infty^2$, and the Lipschitz constant of $F(\delta)$ is $O(\tau)$, lead to the following theorem.

**Theorem 5.3.** *Given any algorithm for optimizing $DMAP_\tau$ and $\epsilon \in R(\tau)$, if the algorithm is guaranteed to achieve $F(\delta^t) - F(\delta^*) \leq \epsilon$ after $O(g(\epsilon))$ iterations, then it is guaranteed to be $\epsilon$ primal optimal, i.e., $P_\tau^* - P_\tau(\tilde{\mu}(\delta^t)) \leq \epsilon$ after $O(g(\frac{\epsilon^2}{\tau}))$ iterations.*[8]

The theorem lets us directly translate dual convergence rates into primal ones. Note that it applies to any algorithm for $DMAP_\tau$ (not only coordinate minimization), and the only property of the algorithm used in the proof is $F(\delta^t) \leq F(0)$ for all $t$. Put in the context of our previous results, any algorithm that achieves $F(\delta^t) - F(\delta^*) \leq \epsilon$ in $t = O(\tau/\epsilon)$ iterations, then it is guaranteed to achieve $P_\tau^* - P_\tau(\tilde{\mu}(\delta^{t'})) \leq \epsilon$ in $t' = O(\tau^2/\epsilon^2)$ iterations.

## 6 Experiments

In this section we evaluate coordinate minimization algorithms on a MAP problem, and compare them to state-of-the-art baselines. Specifically, we compare the running time of greedy coordinate minimization, stochastic coordinate minimization, full gradient descent, and FISTA – an accelerated gradient method [1] (details on the gradient-based algorithms are provided in the supplementary, Section 3). Gradient descent is known to converge in $O\left(\frac{1}{\epsilon}\right)$ iterations while FISTA converges in $O\left(\frac{1}{\sqrt{\epsilon}}\right)$ iterations [1]. We compare the performance of the algorithms on protein side-chain prediction problems from the dataset of Yanover et al. [28]. These problems involve finding the 3D configuration of rotamers given the backbone structure of a protein. The problems are modeled by singleton and pairwise factors and can be posed as finding a MAP assignment for the given model.

Figure 1(a) shows the objective value for each algorithm over time. We first notice that the greedy algorithm converges faster than the stochastic one. This is in agreement with our theoretical analysis. Second, we observe that the coordinate minimization algorithms are competitive with the accelerated gradient method FISTA and are much faster than the gradient method. Third, as Theorem 5.3 predicts, primal convergence is slower than dual convergence (notice the logarithmic timescale). Finally, we can see that better convergence of the dual objective corresponds to better convergence of the primal objective, in both fractional and integral domains. In our experiments the quality of the decoded integral solution (dashed lines) significantly exceeds that of the fractional solution. Although sometimes a fractional solution can be useful in itself, this suggests that if only an integral solution is sought then it could be enough to decode directly from the dual variables.

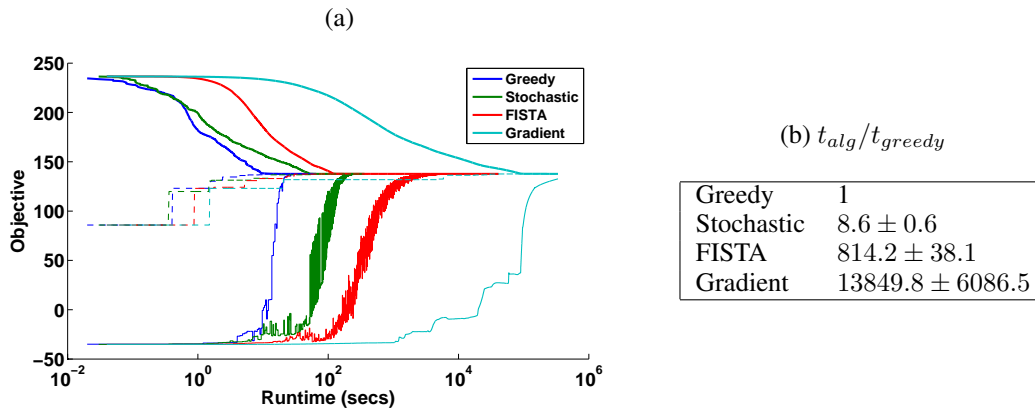

(a)

(b) $t_{alg}/t_{greedy}$

| Greedy | 1 |
| Stochastic | $8.6 \pm 0.6$ |
| FISTA | $814.2 \pm 38.1$ |
| Gradient | $13849.8 \pm 6086.5$ |

Figure 1: Comparison of coordinate minimization, gradient descent, and the accelerated gradient algorithms on protein side-chain prediction task. Figure (a) shows a typical run of the algorithms. For each algorithm the dual objective of Eq. (6) is plotted as a function of execution time. The value (Eq. (4)) of the feasible primal solution of Algorithm 1 is also shown (lower solid line), as well as the objective (Eq. (1)) of the best decoded integer solution (dashed line; those are decoded directly from the dual variables $\delta$). Table (b) shows the ratio of runtime of each algorithm w.r.t. the greedy algorithm. The mean ratio over the proteins in the dataset is shown followed by standard error.

The table in Figure 1(b) shows overall statistics for the proteins in the dataset. Here we run each algorithm until the duality gap drops bellow a fixed desired precision ($\epsilon = 0.1$) and compare the total runtime. The table presents the ratio of runtime of each algorithm w.r.t. the greedy algorithm ($t_{alg}/t_{greedy}$). These results are consistent with the example in Figure 1(a).

## 7 Discussion

We presented the first convergence rate analysis of dual coordinate minimization algorithms on MAP-LP relaxations. We also showed how such dual iterates can be turned into primal feasible iterates and analyzed the rate with which these primal iterates converge to the primal optimum. The primal mapping is of considerable practical value, as it allows us to monitor the distance between the upper (dual) and lower (primal) bounds on the optimum and use this as a stopping criterion. Note that this cannot be done without a primal feasible solution.[9]

The overall rates we obtain are of the order $O(\frac{\tau}{\epsilon})$ for the $DMAP_\tau$ problem. If one requires an $\epsilon$ accurate solution for $PMAP$, then $\tau$ needs to be set to $O(\frac{1}{\epsilon})$ (see Eq. (5)) and the overall rate is $O(\frac{1}{\epsilon^2})$ in the dual. As noted in [8, 18], a faster rate of $O(\frac{1}{\epsilon})$ may be obtained using accelerated methods such as Nesterov's [15] or FISTA [1]. However, these also have an extra factor of $N$ which does not appear in the greedy rate. This could partially explain the excellent performance of the greedy scheme when compared to FISTA (see Section 6).

Our analysis also highlights the advantage of using greedy block choice for MAP problems. The advantage comes from the fact that the choice of block to update is quite efficient since its cost is of the order of the other computations required by the algorithm. This can be viewed as a theoretical reinforcement of selective scheduling algorithms such as Residual Belief Propagation [4].

Many interesting questions still remain to be answered. How should one choose between different block updates (e.g., MSD vs star)? What are lower bounds on rates? Can we use acceleration as in [15] to obtain better rates? What is the effect of adaptive smoothing (see [19]) on rates? We plan to address these in future work.

**Acknowledgments:** This work was supported by BSF grant 2008303. Ofer Meshi is a recipient of the Google Europe Fellowship in Machine Learning, and this research is supported in part by this Google Fellowship.

## Footnotes

[1] Although singleton factors are not needed for generality, we keep them for notational convenience.

[2] We use $\mu$ and $\theta$ to denote vectors consisting of all $\mu$ and $\theta$ values respectively.

[3]Non uniform schedules are also possible. We consider the uniform for simplicity.

[4]The update is presented here in additive form, there is an equivalent absolute form [21].

[5]Expectation is taken with respect to the randomization of blocks.

[6]We also provide a direct proof in the supplementary, Section 2.

[7]This was also used in the residual belief propagation approach [4], which however is less theoretically justified than what we propose here.

[8]We omit constants not depending on $\tau$ and $\epsilon$.

[9]An alternative commonly used progress criterion is to decode an integral solution from the dual variables, and see if its value is close to the dual upper bound. However, this will only work if $PMAP$ has an integral solution and we have managed to decode it.

# References

[1] A. Beck and M. Teboulle. A fast iterative shrinkage-thresholding algorithm for linear inverse problems. *SIAM J. Img. Sci.*, 2(1):183–202, Mar. 2009.

[2] Y. Boykov, O. Veksler, and R. Zabih. Fast approximate energy minimization via graph cuts. In *Proc. IEEE Conf. Comput. Vision Pattern Recog.*, 1999.

[3] D. Burshtein. Iterative approximate linear programming decoding of ldpc codes with linear complexity. *IEEE Transactions on Information Theory*, 55(11):4835–4859, 2009.

[4] G. Elidan, I. Mcgraw, and D. Koller. Residual belief propagation: informed scheduling for asynchronous message passing. In *UAI*, 2006.

[5] A. Globerson and T. Jaakkola. Fixing max-product: Convergent message passing algorithms for MAP LP-relaxations. In J. Platt, D. Koller, Y. Singer, and S. Roweis, editors, *NIPS 20*. MIT Press, 2008.

[6] M. Guignard and S. Kim. Lagrangean decomposition: A model yielding stronger Lagrangean bounds. *Mathematical Programming*, 39(2):215–228, 1987.

[7] T. Hazan and A. Shashua. Norm-product belief propagation: Primal-dual message-passing for approximate inference. *IEEE Transactions on Information Theory*, 56(12):6294–6316, 2010.

[8] V. Jojic, S. Gould, and D. Koller. Fast and smooth: Accelerated dual decomposition for MAP inference. In *Proceedings of International Conference on Machine Learning (ICML)*, 2010.

[9] V. Kolmogorov. Convergent tree-reweighted message passing for energy minimization. *IEEE Transactions on Pattern Analysis and Machine Intelligence*, 28(10):1568–1583, 2006.

[10] A. L. Martins, M. A. T. Figueiredo, P. M. Q. Aguiar, N. A. Smith, and E. P. Xing. An augmented lagrangian approach to constrained map inference. In *ICML*, pages 169–176, 2011.

[11] K. Matusita. On the notion of affinity of several distributions and some of its applications. *Annals of the Institute of Statistical Mathematics*, 19:181–192, 1967. 10.1007/BF02911675.

[12] O. Meshi and A. Globerson. An alternating direction method for dual map lp relaxation. In *ECML PKDD*, pages 470–483. Springer-Verlag, 2011.

[13] O. Meshi, D. Sontag, T. Jaakkola, and A. Globerson. Learning efficiently with approximate inference via dual losses. In *ICML*, pages 783–790, New York, NY, USA, 2010. ACM.

[14] Y. Nesterov. *Introductory Lectures on Convex Optimization: A Basic Course*, volume 87. Kluwer Academic Publishers, 2004.

[15] Y. Nesterov. Smooth minimization of non-smooth functions. *Math. Prog.*, 103(1):127–152, May 2005.

[16] Y. Nesterov. Efficiency of coordinate descent methods on huge-scale optimization problems. Core discussion papers, Universit catholique de Louvain, 2010.

[17] A. Saha and A. Tewari. On the finite time convergence of cyclic coordinate descent methods, 2010. preprint arXiv:1005.2146.

[18] B. Savchynskyy, S. Schmidt, J. Kappes, and C. Schnorr. A study of Nesterov's scheme for lagrangian decomposition and map labeling. *CVPR*, 2011.

[19] B. Savchynskyy, S. Schmidt, J. H. Kappes, and C. Schnörr. Efficient mrf energy minimization via adaptive diminishing smoothing. In *UAI*, 2012.

[20] S. Shalev-Shwartz and A. Tewari. Stochastic methods for l1-regularized loss minimization. *J. Mach. Learn. Res.*, 12:1865–1892, July 2011.

[21] D. Sontag, A. Globerson, and T. Jaakkola. Introduction to dual decomposition for inference. In *Optimization for Machine Learning*, pages 219–254. MIT Press, 2011.

[22] P. Tseng. Convergence of a block coordinate descent method for nondifferentiable minimization 1. *Journal of Optimization Theory and Applications*, 109(3):475–494, 2001.

[23] M. Wainwright, T. Jaakkola, and A. Willsky. MAP estimation via agreement on trees: message-passing and linear programming. *IEEE Transactions on Information Theory*, 51(11):3697–3717, 2005.

[24] M. Wainwright and M. I. Jordan. *Graphical Models, Exponential Families, and Variational Inference*. Now Publishers Inc., Hanover, MA, USA, 2008.

[25] T. Werner. A linear programming approach to max-sum problem: A review. *IEEE Transactions on Pattern Analysis and Machine Intelligence*, 29(7):1165–1179, 2007.

[26] T. Werner. Revisiting the decomposition approach to inference in exponential families and graphical models. Technical Report CTU-CMP-2009-06, Czech Technical University, 2009.

[27] T. Werner. How to compute primal solution from dual one in MAP inference in MRF? In *Control Systems and Computers (special issue on Optimal Labeling Problems in Structual Pattern Recognition)*, 2011.

[28] C. Yanover, T. Meltzer, and Y. Weiss. Linear programming relaxations and belief propagation – an empirical study. *Journal of Machine Learning Research*, 7:1887–1907, 2006.

